# A Reinforcement Learning Theory for Homeostatic Regulation

**Mehdi Keramati**
Group for Neural Theory, LNC, ENS
Paris, France
mohammadmahdi.keramati@ens.fr

**Boris Gutkin**
Group for Neural Theory, LNC, ENS
Paris, France
boris.gutkin@ens.fr

## Abstract

Reinforcement learning models address animal's behavioral adaptation to its changing "external" environment, and are based on the assumption that Pavlovian, habitual and goal-directed responses seek to maximize reward acquisition. Negative-feedback models of homeostatic regulation, on the other hand, are concerned with behavioral adaptation in response to the "internal" state of the animal, and assume that animals' behavioral objective is to minimize deviations of some key physiological variables from their hypothetical setpoints. Building upon the drive-reduction theory of reward, we propose a new analytical framework that integrates learning and regulatory systems, such that the two seemingly unrelated objectives of reward maximization and physiological-stability prove to be identical. The proposed theory shows behavioral adaptation to both internal and external states in a disciplined way. We further show that the proposed framework allows for a unified explanation of some behavioral pattern like motivational sensitivity of different associative learning mechanism, anticipatory responses, interaction among competing motivational systems, and risk aversion.

## 1 Introduction

"Reinforcement learning" and "negative-feedback models of homeostatic regulation" are two control theory-based computational frameworks that have had major contributions in learning and motivation literatures in behavioral psychology, respectively.

Proposing neurobiologically-plausible algorithms, computational theory of reinforcement learning (RL) explains how animals adapt their behavior to varying contingencies in their "external" environment, through persistently updating their estimates of the rewarding value of feasible choices [1]. The teaching signal required for these updates is suggested to be carried by phasic activity of midbrain dopamine neurons [2] projecting to the striatum, where stimulus-response associations are supposed to be encoded. Critically, this theory is built upon one hypothetical assumption: animals behavioral objective is to maximize reward acquisition. In this respect, without addressing the question of how the brain constructs reward, the RL theory gives a normative explanation for how the brains decision making circuitry shapes instrumental responses as a function of external cues, so as the animal to satisfy its reward-maximization objective.

Negative-feedback models of homeostatic regulation (HR), on the other hand, seek to explain regular variabilities in behavior as a function of the internal state of the animal, when the external stimuli are fixed [3, 4]. Homeostasis means maintaining stable of some physiological variables. Correspondingly, behavioral homeostasis refers to corrective responses that are triggered by deviation of

a regulated variable from its hypothetical setpoint. In fact, regulatory systems operate "as if" they aim at defending variables against perturbations. In this sense, existence of an "error signal" (also known as "negative feedback"), defined by the discrepancy between the current and desired internal state, is essential for any regulatory mechanism.

Even though the presence of this negative feedback in controlling motivated behaviors is argued to be indisputable for explaining some behavioral and physiological facts [3], three major difficulties have raised criticism against negative-feedback models of behavioral homeostasis [3, 5]: (1) Anticipatory eating and drinking can be elicited in the absence of physiological depletion [6], supposedly in order to prevent anticipated deviations in future. In more general terms, motivated behaviors are observed to be elicited even when no negative feedback is detectable. (2) Intravenous (and even intragastric, in some cases) injection of food is not motivating, even though it alleviates deviation of the motivational state from its homeostatic setpoint. For example, rats do not show motivation, after several learning trials, to run down an alley for intragastric feeding by milk, whereas they quickly learn to run for normal drinking of milk [7]. These behavioral patterns are used as evidence to argue that maintaining homeostasis (reducing the negative feedback) is not a behavioral objective. In contrast, taste (and other sensory) information of stimuli is argued to be the key factor for their reinforcing properties [5]. (3) The traditional homeostatic regulation theory simply assumes that the animal knows how to translate its various physiological deficits into the appropriate behaviors. In fact, without taking into account the contextual state of the animal, negative feedback models only address the question of whether or not the animal should have motivation toward a certain outcome, without answering how the outcome can be achieved through a series of instrumental actions.

The existence of these shortcomings calls for rethinking the traditional view toward homeostatic regulation theory. We believe that these shortcomings, as well as the weak spot of RL models in not taking into account the internal drive state of the animal, all arise from lack of an integrated view toward learning and motivation. We show in this paper that a simple unified computational theory for RL and HR allows for explaining a wide range of behavioral patterns including those mentioned above, without decreasing the explanatory power of the two original theories. We further show that such a unified theory can satisfy the two objectives of reward maximization and deviation minimization at the same time.

## 2    The model

The term "reward" (with many equivalents like reinforcer, motivational salience, utility, etc.) has been at the very heart of behavioral psychology since its foundation. In purely behavioral terms, it refers to a stimulus delivered to the animal after a response is made, that increases the probability of making that response in the future. RL theory proposes algorithms for how different learning systems can adapt agent's responses to varying external conditions in order to maximize the acquisition of reward. For this purpose, RL algorithms try to learn, via experiences, the sum of discounted rewards expected to be received after taking a specific action ($a_t$), in a specific state ($s_t$), onward.

$$V(s_t, a_t) = E\left[r_t + \gamma r_{t+1} + \gamma^2 r_{t+2} + \ldots | s_t, a_t\right] = E\left[\sum_{i=t}^{\infty} \gamma^{i-t} r_i | s_t, a_t\right] \tag{1}$$

$0 \leq \gamma \leq 1$ discounts future rewards. $r_t$ denotes the rewarding value of the outcome the animal receives at time $t$, which is a often set to a "fixed" value that can be either positive or negative, depending on whether the corresponding stimulus is appetitive or aversive, respectively. However, animals motivation for outcomes is not fixed, but a function of their internal state: a food pellet is more rewarding to a hungry, than a sated rat. In fact, the internal state (also referred to as drive, or motivational state) of the animal affects the reinforcing value of a constant external outcome.

As an attempt to identify the nature of reward and its relation to drive states, neo-behaviorists like Hull [8], Spence, and Mowrer have proposed the "drive reduction" theory of motivation. According to this theory, one primary mechanism underlying reward is drive reduction. In terms of homeostatic

regulation theory, reward is defined as a stimulus that reduces the discrepancy between the current and the desired drive state of the animal, i.e, food pellet is rewarding to a hungry rat because it fulfills a physiological need.

To capture this idea formally, let's $H_t = \{h_{1,t}, h_{2,t}, .., h_{N,t}\}$ denote the physiological state of the animal at time $t$, and $H^* = \{h_1^*, h_2^*, .., h_N^*\}$ as the homeostatic setpoint. As a special case, figure 1 shows a simplified system where food and water constitute all the physiological needs. This model can obviously be extended, without loss of generality, to cover other homeostatically regulated drives, as well as more detailed aspects of feeding like differential drives for carbohydrate, sodium, calcium, etc. A drive function can then be defined on this space as a mapping from physio-

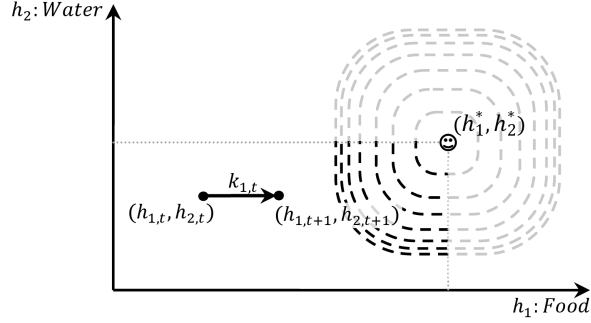

Figure 1: An examplary homeostatic space, with food and water as regulated physiological needs

logical state to motivation:

$$d(H_t) = \sqrt[m]{\sum_{i=1}^{N} |h_i^* - h_{i,t}|^n} \tag{2}$$

This drive function is in fact a distance function (Euclidian distance when $m = n = 2$) that creates some quasi-circle iso-drive curves centred around the homeostatic setpoint. The homeostatic space, as a multidimensional metric space, is a hypothetical construct that allow us to explain a wide range of behavioral and physiological evidence in a unified framework. Since animals physiological states are most often below the setpoint, our focus in this paper is mostly on the quarter of the homeostatic space that is below the homeostatic setpoint.

Having defined drive, we can now provide a formal definition for reward, based on the drive reduction theory. Assume that as the result of some actions, the animal receives an outcome at time $t$ that contains $k_{i,t}$ units of the constituent $h_i$, for each $i \in \{1, 2, .., N\}$. $K_t$ can be defined as a row vector with entries $(k_{i,t} : i \in \{1, 2, .., N\})$. Consumption of that outcome will result in a transition of the physiological state from $H_t$ to $H_{t+1} = H_t + K_t$, and consequently, a transition of the drive state from $d(H_t)$ to $d(H_t + K_t)$. For example, figure 1 shows the transition resulted from taking $k_1$ units of food. Accordingly, the rewarding value of that outcome can be defined as the consequent reduction in the drive function:

$$r(H_t, K_t) = d(H_t) - d(H_{t+1}) = d(H_t) - d(H_t + K_t) \tag{3}$$

This drive-reduction definition of reward is the central element in our proposed framework that will bridge the gap between regulatory and reward learning systems.

## 3 Behavioral plausibility of the reward function

Before discussing how the reward defined in equation 3 can be used by associative learning mechanisms (RL theory), we are interested in this section to show that the functional form proposed for the reward function allows for several behavioral phenomena to be explained in a unified framework.

For all $m > n > 2$, the rewarding value of an outcome consisting of only one constituent, $K_t = (0, 0, .., k_{j,t}, .., 0)$ ,when the animal is in the motivational state $H_t$, will have the four below properties. Even though these properties are written for the cases that the need state remains below the setpoint, as the drive function is symmetric in respect to the setpoint, similar result can be derived for other three quarters.

### 3.1 Reward value increases with the dose of outcome

The reward function is an increasing function of the magnitude of the outcome (e.g., number of food pellets); i.e. the bigger the outcome is, the more rewarding value it will have. It is straightforward to show that:

$$\frac{dr(H_t, K_t)}{dk_{j,t}} > 0 \quad : \quad \text{for } k_{j,t} > 0 \tag{4}$$

Supporting this property of the drive function, it is shown in progressive ratio schedules that rats maintain higher breakpoints when reinforced with bigger appetitive outcomes, reflecting higher motivation toward them [9, 10].

### 3.2 Excitatory effect of deprivation level

Increasing the deprivation level of the animal will increase the reinforcing strength of a constant dose of a corresponding outcome, i.e. a single pellet of food is more rewarding to a hungry, than a sated rate.

$$\frac{dr(H_t, K_t)}{d|h_j^* - h_{j,t}|} > 0 \quad : \quad \text{for } k_{j,t} > 0 \tag{5}$$

Consistently, the level of food deprivation in rats is demonstrated to increases the breakpoint in progressive ratio schedules [10].

### 3.3 Inhibitory effect of the irrelevant drive

A large body of behavioral experiments shows that deprivation level for one need has an inhibitory effect on the reinforcing value of outcomes that satisfy irrelevant needs. (see [11] for review). It is well known that high levels of irrelevant thirst impairs Pavlovian as well as instrumental responses for food during both acquisition and extinction. Reciprocally, food deprivation is demonstrated to suppress Pavlovian and instrumental water-related responses. As some other examples, increased calcium appetite is shown to reduce appetite for phosphorus, and increased level of hunger is demonstrated to inhibit sexual behavior. It is straightforward to show that the specific functional form proposed for the drive function can capture this inhibitory-like interaction between irrelevant drives:

$$\frac{dr(H_t, K_t)}{d|h_i^* - h_{i,t}|} < 0 \quad : \quad \text{for all } i \neq j \text{ and } k_{j,t} > 0 \tag{6}$$

Intuitively, one does not play chess, or even search for sex, with an empty stomach. This behavioral pattern can be interpreted as competition among different motivational systems (different dimensions of the homeostatic space), and is consistent with the notion of "complementary" relation between some goods in economics, like between cake and coffee. Each of two complement goods are highly rewarding, only when the other one is also available; and taking one good when the other one is lacking, is not that rewarding.

### 3.4 Risk aversion

Risk aversion, as a fundamental concept in both psychology and economics, and supported by a wealth of behavioral experiments, is defined by reluctance of individuals to select choices with uncertain outcomes, compared to choices with more certain payoffs, even when the expected payoff of the former is higher than that of the latter. It is easy to show that a concave reward (utility) function

(in respect to the quantity of the outcome), is equivalent to risk aversion. It is due to this feature that concavity of the utility function has become a fundamental assumption in the microeconomic theory. The proposed form of the drive function can capture risk-aversion:

$$\frac{d^2 r(H_t, K_t)}{dk_{j,t}^2} < 0 \quad : \quad \text{for } k_{j,t} > 0 \tag{7}$$

It is noteworthy that as $K_t$ consists of substances that fulfil physiological needs, one should be careful in extending this and other features of the model to the case of monetary rewards, or any other type of stimuli that does not seem to have a corresponding regulatory system (like social rewards, novelty-induced reward, etc.).

It is clear that the four mentioned properties of the reward function depend on the specific functional form adopted for the drive function (equation 2). In fact, the drive-reduction definition of reward allows for the validity of the form of drive function to be experimentally tested in behavioral tasks.

## 4 Homeostatic control through learning mechanisms

Despite significant day-to-day variations in energy expenditure and food availability, the homeostatic regulation system involved in control of food-intake and energy balance is a remarkably precise system. This adaptive nature of homeostatic regulation system can be achieved by employing the animals learning mechanisms, which are capable of adopting flexible behavioral strategies to cope with changes in the environmental conditions. In this section we are interested in looking at behavioral and theoretical implications of interaction between homeostatic regulation and reward learning systems.

One theoretical implication of the proposed definition of reward is that it reconciles the RL and homeostatic regulation theories, in terms of normative assumptions behind them. In more precise words, if any reward learning system, like RL algorithms, uses the drive-reduction definition of reward proposed in equation 3, then a behavioral policy that maximizes the sum of discounted rewards is at the same time minimizing the sum of discounted drives (or sum of discounted deviations from the setpoint), and vice versa (see Supplementary Methods: Proposition 1). In this respect, reward maximization can be seen as just means that guide animal's behavior toward satisfying the more basic objective of maintaining homeostasis.

Since the reward function defined by equation 3 depends on the internal state and thus, is non-stationary, some appropriate adjustments in the classical RL algorithms that try to find an optimal policy must be thought of. One straightforward solution is to define an augmented Markov decision process (MDP) implied by the cross-product of internal and external states, an then use a variant of RL algorithms to learn action-values in that problem-space. Consistent with this view, some behavioral studies show that internal state can work in the same way that external state works. That is, animals are able to acquire responses conditioned upon some certain motivational states (e.g. motivational state induced by benzodiazepine agonist) [11]. Although this approach, in theory, guarantees convergence to the optimal policy, high dimensionality of the resulting state-space makes learning rather impossible in practice. Moreover, since the next external state only depends on the current external but not internal state, such an augmented MDP will have significant redundant structure. From a machine learning point of view, as argued in [12], an appropriate function approximator specifically designed to take advantage of such a structure can be used to reduce state-space dimensionality.

Beside this algorithm-independent expansion of state-space argued above, the proposed definition of reward provides an opportunity for discussing how different associate learning systems in the brain take the animal's internal state into account. Here we discuss motivational sensitivity in habitual (hypothesized to use a model-free RL, like temporal difference algorithm [13]), goal-directed (hypothesized to use a model-based RL [13]), and Pavlovian learning systems.

A model-based RL algorithm learns the state-transition matrix (action-outcome contingencies), as well as the reward function (the incentive value of each outcome), and then uses them to compute the value of possible actions in a given state, using Bellman optimality equation[1]. In our framework, as long as a model-based algorithm is involved in decision making, all that the animal needs to do when its internal state shifts is to update the incentive value (reward function) of each potential outcome. The reward function being updated, the animal will be able to take the optimal policy, given that the state-transition function is learned completely. In order to update the reward function, one way is that the animal re-experiences outcomes in its new motivational state. This way seems to be how the goal-directed system works, since re-exposure is demonstrated to be necessary in order for the animals' behavior to be sensitive to changes in motivational state [11]. The second way is to update the reward function without re-exposure, but through directly estimating the drive-reduction effect of outcomes in the current motivational state, using equation 3. This way seems to be how the Pavlovian system works, since numerous experiments show that Pavlovian conditioned or unconditioned responses are sensitive to the internal state in an outcome-specific way, without re-exposure being required [11]. This Pavlovian-type behavioral adaptation is observed even when the animal has never experienced the outcome in its current motivational state during its whole life. This shows that animals are able to directly estimate the drive-reduction effect of at least some certain types of outcomes. Furthermore, the fact that this motivational sensitivity is outcome-specific shows that the underlying associative learning structure is model-based (i.e. based on learning the causal contingencies between events).

The habitual system (hypothesized to use a model-free RL[13]) is demonstrated, through outcome-devaluation experiments, that cannot directly (i.e. without new learning) adapt the animal's behavioral response after shifts in the motivational state [11]. It might be concluded from this observation that action-values in the habitual system only depend on the internal state of the animal during the course of learning (past experiences), but not performance. This is consistent with the information storage scheme in model-free RL, where cached action-values lose connection with the identity of the outcomes. Thus, the habitual system doesn't know what specific action-values should be updated when a novel internal state is being experienced.

However, it has been argued by some authors [14] that even though habitual responses are insensitive to sensory-specific satiety, they are sensitive to motivational manipulations in a general, outcome-independent way. Note that the previous form of motivational insensitivity concerns lack of behavioral adaptation in an outcome-specific way (for example, lack of greater preference for food-seeking behavior, compared to water-seeking behavior, after a shift from satiety to hunger state). Borrowing from the Hullian concept of "generalized drive" [8], it has been proposed that transition from an outcome-sated to an outcome-deprived motivational state will energize all pre-potent habitual responses, irrespective of whether or not those actions result in that certain outcome [14]. For example, a motivational shift from satiety to hunger will result in energization of both food-seeking and water-seeking habitual responses.

Taking a normative perspective, we argue here that this outcome-independent energizing effect is an approximate way of updating the value of state-action pairs when the motivational state shifts instantaneously. Assuming that the animal is trained under the fixed internal state $H_0$, and then tested in a novel internal state $H_1$, the cached values in the habitual system can be approximated in the new motivational state by:

$$Q_1(s,a) = \frac{d(H_1)}{d(H_0)}.Q_0(s,a) \ : \ \text{ for all state-action pairs} \tag{8}$$

Where $Q_0(s,a)$ represents action-values learned by the habitual system after the training period. According to this update rule, all the prepotent actions will get energized if deviation from the homeostatic setpoint increases in the new internal state, whether or not the outcome of those actions are more desired in the new state. Proposition 2 (see Supplementary Methods) shows that this update rule is a perfect approximation only when $H_1 = c.H_0$, where $c \geq 0$. This means that the energizing effect will result in rational behavior after motivational shift, only when the new internal state is an

amplification or abridge of the old internal state in all the dimension of the homeostatic space, with equal magnitude. Since the model-free system does not learn the causal model of the environment, it cannot show motivational sensitivity in an outcome-specific way, and this general energizing effect is the best approximate way to react to motivational manipulations.

## 4.1 Anticipatory responses

The notion of predictive homeostasis [3, 4], as opposed to the classical reactive homeostasis where the existence of negative feedback (physiological depletion) is essential, suggests that through anticipating future needs, individuals make anticipatory responses in order to prevent deviation of regulated variables from their setpoints. Anticipatory eating and drinking, as two examples, are defined by taking food and water in the absence of any detectable decrease in the corresponding physiological signals.

Although it is quite clear that explaining such behavioral patterns requires integrating homeostatic mechanisms and learning (as predictive) processes, to our knowledge, no well-defined mechanism has been proposed for it so far. We show here, through an example, how the proposed model can reconcile anticipatory responses to the homeostatic regulation theory. Temperature regulation provides a clear example for predictive homeostasis.

Body temperature, which has long been a topic of interest in the homeostatic regulation literature, is shown to increase back to its normal level by shivering, after the animal is placed into a cold place. Interestingly, cues that predict being placed into a cold place induce anticipatory shivering and result in the body temperature to go above the normal temperature of the animal [15]. Similarly, cues that predict receiving a drug that decreases body temperature is shown to have the same effect. This behavior is interpreted as an attempt by the animal to alleviate the severity of deviation from the setpoint.

To model this example, lets assume that $x^*$ is the normal temperature of the animal, and that putting the animal in the cold place will result in a decrease of $l_x$ units in the body temperature. Furthermore, assume that when presented with the coldness-predicting cue ($S_C$), the animal chooses how much to shiver and thus, increases its body temperature by $k_x$ units. In a scenario like this, after observing the coldness-predicting cue, the animals temperature will shift from $x^*$ to $x^* + k_x$, as a result of anticipatory shivering. Assuming that the animal will then experience coldness after a delay of one time-unit, its temperature will transit from $x^* + k_x$ to $x^* + k_x - l_x$. The rewarding value of this transition will be discounted with the rate of $\gamma$. Finally, by assuming that after one more time-unit the body temperature will go back to the normal level, $x^*$, the animal will receive another reward, discounted with the rate of $\gamma^2$. The sum of discounted rewards can be written as below:

$$V(S_C, k_X) = [d(x^*) - d(x^* + k_X)] + \gamma.[d(x^* + k_X) - d(x^* + k_X - l_X)] + \gamma^2.[d(x^* + k_X - l_X) - d(x^*)] \tag{9}$$

Proposition 3 (see Supplementary Methods) shows that the optimal strategy for maximizing the sum of discounted rewards in this scenario is when $k_x = l_x/2$, assuming that the discount factor is not equal to one, but is sufficiently close to it. In fact, the model predicts that the best strategy is to perform anticipatory shivering to the extent that keep it as close as possible to the setpoint: turning around but close to the setpoint is preferred to getting far from it and coming back. In fact this example, which can be easily generalized to anticipatory eating and drinking, shows that when learning mechanisms play a regulatory role, it is not only the initial and final motivational states of a policy that matters, but also the trajectory of the motivational state in the homeostatic space through that policy matters. It is in fact due to discounting future rewards. If the discount factor is one, then regardless of the trajectory of the motivational state, the sum of rewards for all policies that start from a certain homeostatic point and finish at another point, will be equal. In that case, the sum of rewards for all values of $k_x$ in the anticipatory shivering example will be zero and thus, anticipatory strategies will not be preferred to a reactive-homeostasis strategy. In fact, the model predicts that decreasing animal's discount rate (e.g. through pharmacological agents known to modulate discount rate) should increase the probability that the animal show an anticipatory response, and vice versa.

# 5 Discussion

Despite considerable differences, some common principles are argued to govern all motivational systems. we have proposed a model that captures some of these commonalities between homoestatically regulated motivational systems. We showed how the rewarding value of an outcome can be computed as a function of the animal's internal state and the constituting elements of the outcome, through the drive reduction equation. We further demonstrated how this computed reward can be used by different learning mechanisms to form appropriate Pavlovian or instrumental associations.

However, it should be noted that concerning food (and other ingestible) outcomes, the basic form of the drive-reduction definition of reward (equation 2) has four interrelated problems, some of them used traditionally to criticize the drive reduction theory: (1) Post-digestive nutritive feedback of food, defined by the drive reduction equation, might occur hours after ingestion. Such a long delay between an instrumental response and its consequent drive reduction effect (reward) would make it difficult for an appropriate association to be established. In fact, according to the temporal contiguity rule of associative learning, unconditioned stimuli must follow immediately the conditioned stimuli, for an association to be established between them. (2) Dopamine neurons, which are supposed to carry reward learning signal, are demonstrated to show instantaneous burst activity in response to unexpected food rewards, without waiting for the food to be digested, and drive to be reduced [16]. (3) Intravenous injection (and intragastric intubation, in some cases) of food is not rewarding, even though its drive reduction effect is equal to when that food is ingested orally. As mentioned before, oral ingestion of the same outcome is shown to have significant reinforcing effect [7]. (4) Palatable foods have reinforcing effect, even when they do not have any nutritional value (i.e. they do not reduce any physiological need) [16]. Making the story even more complicated, this taste-dependent rewarding value of food is demonstrated to be modulated by not only the approximated nutritional content of the food, but also the internal physiological state of the animal [16].

However, assuming that taste and other sensory properties of a food outcome give an estimate, $\hat{k}_{1,t}$, of its true nutritional content, $k_{1,t}$, the rewarding effect of food can be approximated by equation 3, as soon as the food is sensed, or taken. This association between sensory information and post-ingestive effects of food might have been established through learning, or evolution. This simple plausible assumption clearly resolves the four problems listed above for the classical notion of drive reduction. It explains that gastric, olfactory, or visual information of food is necessary for its reinforcing effect to be induced and thus, intravenous injection of food is not reinforcing due to lack of appropriate sensory information. Moreover, there is no delay in this mechanism between food intake and its drive-reduction effect and therefore, Dopamine neurons can respond instantaneously. Finally, as equation 3 predicts, this taste-dependent rewarding value is modulated by the motivational state of the animal.

Previous computational accounts in the psychological literature attempting to incorporate internal-state dependence of motivation into the RL models use ad hoc addition or multiplication of drive state with the a priori reward magnitude [17, 13]. In the machine learning literature, among others, Bersini [18] uses an RL model where the agent gets punished if its internal state transgresses a predefined viability zone. Simulation results show that such a setting motivates the agent to maintain its internal variables in a bounded zone. A more recent work [12] also uses RL framework where reward is generated by drive difference. It is demonstrated that this design allows the agent to make a balance in satisfying different drives. Apart from physiological and behavioral plausibility, the theoretical novelty of our proposed framework is in formalizing the hypothetical concept of drive, as a mapping from physiological to motivational state. This has allowed the model to show analytically that reward maximization and deviation minimization can be seen as two sides of the same coin.

# 6 Acknowledgements

MK and BG are supported by grants from Frontiers du Vivant, the French MESR, CNRS, INSERM, ANR, ENP and NERF.

## References

[1] R. S. Sutton and A. G. Barto. *Reinforcement Learning: An Introduction*. MIT Press, Cambridge, 1998.

[2] W. Schultz, P. Dayan, and P. R. Montague. A neural substrate of prediction and reward. *Science*, 275(5306):1593–1599, 1997.

[3] F. M. Toates. *Motivational Systems*. Problems in the behavioral sciences. Cambridge University Press, New York, 1986.

[4] J. E. R. Staddon. *Adaptive behavior and learning*. Cambridge University Press, New York, 1983.

[5] K. C. Berridge. Motivation concepts in behavioral neuroscience. *Physiol Behav*, 81(2):179–209, 2004.

[6] S.C. Woods and R.J. Seeley. Hunger and energy homeostasis. In C. R. Gallistel, editor, *Volume 3 of Steven's Handbook of Experimental Psychology: Learning, Motivation, and Emotion*, pages 633–68. Wiley, New York, third edition, 2002.

[7] N. E. Miller and M. L. Kessen. Reward effects of food via stomach fistula compared with those of food via mouth. *J Comp Physiol Psychol*, 45(6):555–564, 1952.

[8] C. L. Hull. *Principles of behavior: an introduction to behavior theory*. The Century psychology series. Appleton-Century-Crofts, New York, 1943.

[9] P. Skjoldager, P. J. Pierre, and G. Mittleman. Reinforcer magnitude and progressive ratio responding in the rat: Effects of increased effort, prefeeding, and extinction. *Learn motiv*, 24(3):303–343, 1993.

[10] W. Hodos. Progressive ratio as a measure of reward strength. *Science*, 134:943–944, 1961.

[11] A. Dickinson and B. W. Balleine. The role of learning in motivation. In C. R. Gallistel, editor, *Volume 3 of Steven's Handbook of Experimental Psychology: Learning, Motivation, and Emotion*, pages 497–533. Wiley, New York, third edition, 2002.

[12] George Konidaris and Andrew Barto. An adaptive robot motivational system. In *Proceedings of the 9th international conference on Simulation of adaptive behavior : from animals to animats 9*, pages 346–356, 2006.

[13] N. D. Daw, Y. Niv, and P. Dayan. Uncertainty-based competition between prefrontal and dorsolateral striatal systems for behavioral control. *Nat Neurosci*, 8(12):1704–11, 2005.

[14] Y. Niv, D. Joel, and P. Dayan. A normative perspective on motivation. *Trends Cogn Sci*, 10(8):375–381, 2006.

[15] J. G. Mansfield, R. S. Benedict, and S. C. Woods. Response specificity of behaviorally augmented tolerance to ethanol supports a learning interpretation. *Psychopharmacology*, 79(2-3):94–98, 1983.

[16] L. H. Schneider. Orosensory self-stimulation by sucrose involves brain dopaminergic mechanisms. *Ann. N. Y. Acad. Sci*, 575:307–319, 1989.

[17] J. Zhang, K. C. Berridge, A. J. Tindell, K. S. Smith, and J. W. Aldridge. A neural computational model of incentive salience. *PLoS Comp Biol*, 5(7), 2009.

[18] Hugues Bersini. Reinforcement learning for homeostatic endogenous variables. In *Proceedings of the third international conference on Simulation of adaptive behavior : from animals to animats 3*, page 325333, Brighton, United Kingdom, 1994. MIT Press. ACM ID: 189936.

